# Nonnegative Sparse PCA

**Ron Zass   and   Amnon Shashua** *

## Abstract

We describe a nonnegative variant of the "Sparse PCA" problem. The goal is to create a low dimensional representation from a collection of points which on the one hand maximizes the variance of the projected points and on the other uses only parts of the original coordinates, and thereby creating a sparse representation. What distinguishes our problem from other Sparse PCA formulations is that the projection involves only nonnegative weights of the original coordinates — a desired quality in various fields, including economics, bioinformatics and computer vision. Adding nonnegativity contributes to sparseness, where it enforces a partitioning of the original coordinates among the new axes. We describe a simple yet efficient iterative coordinate-descent type of scheme which converges to a local optimum of our optimization criteria, giving good results on large real world datasets.

## 1  Introduction

Both nonnegative and sparse decompositions of data are desirable in domains where the underlying factors have a physical interpretation: In economics, sparseness increases the efficiency of a portfolio, while nonnegativity both increases its efficiency and reduces its risk [7]. In biology, each coordinate axis may correspond to a specific gene, the sparseness is necessary for finding focalized local patterns hidden in the data, and the nonnegativity is required due to the robustness of biological systems – where observed change in the expression level of a specific gene emerges from either positive or negative influence, rather than a combination of both which partly cancel each other [1]. In computer vision, coordinates may correspond to pixels, and nonnegative sparse decomposition is related to the extraction of relevant parts from images [10]; and in machine learning sparseness is closely related to feature selection and to improved generalization in learning algorithms, while nonnegativity relates to probability distributions.

Principal Component Analysis (PCA) is a popular wide spread method of data decomposition with applications throughout science and engineering. The decomposition performed by PCA is a linear combination of the input coordinates where the coefficients of the combination (the principal vectors) form a low-dimensional subspace that corresponds to the direction of maximal variance in the data. PCA is attractive for a number of reasons. The maximum variance property provides a way to compress the data with minimal information loss. In fact, the principal vectors provide the closest (in least squares sense) linear subspace to the data. Second, the representation of the data in the projected space is uncorrelated, which is a useful property for subsequent statistical analysis. Third, the PCA decomposition can be achieved via an eigenvalue decomposition of the data covariance matrix.

Two particular drawbacks of PCA are the lack of sparseness of the principal vectors, i.e., all the data coordinates participate in the linear combination, and the fact that the linear combination may mix both positive and negative weights, which might partly cancel each other. The purpose of our work is to incorporate both nonnegativity and sparseness into PCA, maintaining the maximal variance property of PCA. In other words, the goal is to find a collection of sparse nonnegative principal

vectors spanning a low-dimensional space that preserves as much as possible the variance of the data. We present an efficient and simple algorithm for Nonnegative Sparse PCA, and demonstrate good results over real world datasets.

## 1.1 Related Work

The desire of adding a sparseness property to PCA has been a focus of attention in the past decade starting from the work of [8] who applied axis rotations and component thresholding to the more recent computational techniques SCoTLASS $L_1$ norm approach [9], elastic net $L_1$ regression SPCA [14], DSPCA based on relaxing a hard cardinality cap constraint with a convex approximation [2], and most recently the work of [12] which applies post-processing renormalization steps to improve any approximate solution, in addition to two different algorithms that search for the active coordinates of the principal component based on spectral bounds. These references above can be divided into two paradigms: (i) adding $L_1$ norm terms to the PCA formulation as it is known that $L_1$ approximates $L_0$ much better than $L_2$, (ii) relaxing a hard cardinality ($L_0$ norm) constraint on the principal vectors. In both cases the orthonormality of the principal vector set is severely compromised or even abandoned and it is left unclear to what degree the resulting principal basis explains most of the variance present in the data.

While the above methods do not deal with nonnegativity at all, other approaches focus on nonnegativity but are neutral to the variance of the resulting factors, and hence recover parts which are not necessarily informative. A popular example is the Nonnegative Matrix Factorization (NMF) [10] and the sparse versions of it [6, 11, 5, 4] that seek the best reconstruction of the input using nonnegative (sparse) prototypes and weights.

We start with adding nonnegativity to PCA. An interesting direct byproduct of nonnegativity in PCA is that the coordinates split among the principal vectors. This makes the principal vectors disjoint, where each coordinate is non-zero in at most one vector. We can therefore view the principal vectors as parts. We then relax the disjoint property, as for most applications some overlap among parts is desired, allowing some overlap among the principal vectors. We further introduce a "sparseness" term to the optimization criterion to cover situations where the part (or semi-part) decomposition is not sufficient to guarantee sparsity (such as when the dimension of the input space far exceeds the number of principal vectors).

The structure of the paper is as follows: In Sections 2 and 3 we introduce the formulation of Nonnegative Sparse PCA. An efficient coordinate descent algorithm for finding a local optimum is derived in Section 4. Our experiments in Section 5 demonstrate the effectiveness of the approach on large real-world datasets, followed by conclusions in Section 6.

## 2 Nonnegative (Semi-Disjoint) PCA

To the original PCA, which maximizes the variance, we add nonnegativity, showing that this addition alone ensures some sparseness by turning the principal vectors into a disjoint set of vectors, meaning that each coordinate is non-zero in at most one principal vector. We will later relax the disjoint property, as it is too excessive for most applications.

Let $\mathbf{x}_1, ..., \mathbf{x}_n \in R^d$ form a zero mean collection of data points, arranged as the columns of the matrix $X \in R^{d \times n}$, and $\mathbf{u}_1, ..., \mathbf{u}_k \in R^d$ be the desired principal vectors, arranged as the columns of the matrix $U \in R^{d \times k}$. Adding a nonnegativity constraint to PCA gives us the following optimization problem:

$$\max_U \frac{1}{2} \|U^T X\|_F^2 \quad s.t. \quad U^T U = I, \quad U \geq 0 \tag{1}$$

where $\|A\|_F^2 = \sum_{ij} a_{ij}^2$ is the square Frobenius norm. Clearly, the combination of $U^T U = I$ and $U \geq 0$ entails that $U$ is disjoint, meaning that each row of $U$ contains at most one non-zero element. While having disjoint principal component may be considered as a kind of sparseness, it is too restrictive for most problems. For example, a stock may be a part of more than one sector, genes are typically involved in several biological processes [1], a pixel may be a shared among several image parts, and so forth. We therefore wish to allow some overlap among the principal vectors. The degree of coordinate overlap can be represented by an orthonormality distance measure which

is nonnegative and vanishes iff $U$ is orthonormal. The function $\|I - U^T U\|_F^2$ is typically used in the literature (cf. [13], pg. 275–277) as a measure for orthonormality and the relaxed version of eqn. 1 becomes,

$$\max_U \frac{1}{2}\|U^T X\|_F^2 - \frac{\alpha}{4}\|I - U^T U\|_F^2 \quad s.t. \quad U \geq 0 \tag{2}$$

where $\alpha > 0$ is a balancing parameter between reconstruction and orthonormality. We see that the tradeoff for relaxing the disjoint property of Nonnegative PCA is also to relax the maximum variance property of PCA — the constrained optimization tries to preserve the variance when possible but allows to tradeoff higher variance with some degree of coordinate overlap among the principal vectors. Next, we add sparseness to this formulation.

## 3  Nonnegative Sparse PCA (NSPCA)

While semi-disjoint principal components can be considered sparse when the number of coordinates is small, it may be too dense when the number of coordinates highly exceeds the number of principal vectors. In such case, the average number of non-zero elements per principal vector would be high. We therefore consider minimizing the number of non-zero elements directly, $\|U\|_{L_0} = \sum_{i=1}^k \sum_{j=1}^n \delta_{u_{ij}}$, where $\delta_x$ equals one if $x$ is non-zero and zero otherwise. Adding this to the criteria of eqn. 2 we have,

$$\max_U \frac{1}{2}\|U^T X\|_F^2 - \frac{\alpha}{4}\|I - U^T U\|_F^2 - \beta\|U\|_{L_0} \quad s.t. \quad U \geq 0$$

where $\beta \geq 0$ controls the amount of additional sparseness required. The $L_0$ norm could be relaxed by replacing it with a $L_1$ term and since $U$ is nonnegative we obtain the relaxed sparseness term: $\|U\|_{L_1} = \mathbf{1}^T U \mathbf{1}$, where $\mathbf{1}$ is a column vector with all elements equal to one. The relaxed problem becomes,

$$\max_U \frac{1}{2}\|U^T X\|_F^2 - \frac{\alpha}{4}\|I - U^T U\|_F^2 - \beta \mathbf{1}^T U \mathbf{1} \quad s.t. \quad U \geq 0 \tag{3}$$

## 4  Algorithm

For certain values of $\alpha$ and $\beta$, solving the problem of eqn. 3 is NP-hard. For example, for large enough values of $\alpha$ and for $\beta = 0$ we obtain the original problem of eqn. 1. This is a concave quadratic programming, which is an NP-hard problem [3]. It is therefore unrealistic to look for a global solution of eqn. 3, and we have to settle with a local maximum.

The objective of eqn. 3 as a function of $u_{rs}$ (the $s$ row of the $u_r$ column vector) is,

$$f(u_{rs}) = -\frac{\alpha}{4}u_{rs}^4 + \frac{c_2}{2}u_{rs}^2 + c_1 u_{rs} + const \tag{4}$$

where $const$ stands for terms that do not depend on $u_{rs}$ and,

$$c_1 = \sum_{i=1,i\neq s}^d a_{si}u_{ri} - \alpha \cdot \sum_{i=1,i\neq r}^k \sum_{j=1,j\neq s}^d u_{rj}u_{ij}u_{is} - \beta,$$

$$c_2 = a_{ss} + \alpha - \alpha \cdot \sum_{i=1,i\neq s}^d u_{ri}^2 - \alpha \cdot \sum_{i=1,i\neq r}^k u_{is}^2$$

where $A = XX^T$. Setting the derivative with respect to $u_{rs}$ to zero we obtain a cubic equation,

$$\frac{\partial f}{\partial u_{rs}} = -\alpha u_{rs}^3 + c_2 u_{rs} + c_1 = 0 \tag{5}$$

Evaluating eqn. 4 for the nonnegative roots of eqn. 5 and zero, the nonnegative global maximum of $f(u_{rs})$ can be found (see Fig. 1). Note that as $u_{rs}$ approaches $\infty$ the criteria goes to $-\infty$, and since the function is continues a nonnegative maximum must exist. A coordinate-descent scheme for updating each entry of $U$ one following the other would converge to a local maximum of the

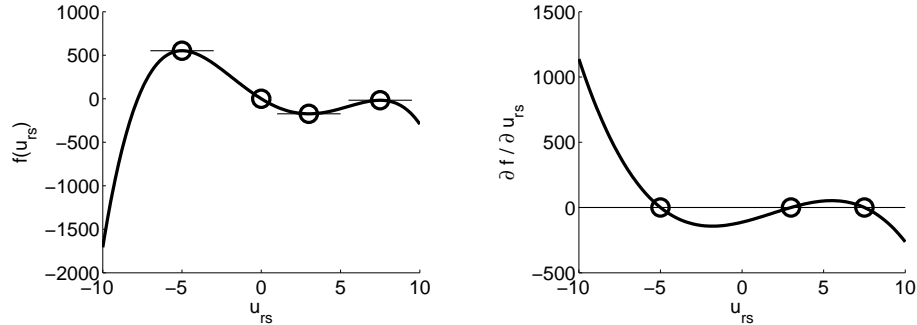

Figure 1: A $4^{th}$ order polynomial (left) and its derivative (right). In order to find the global nonnegative maximum, the function has to be inspected at all nonnegative extrema (where the derivative is zero) and at $u_{rs} = 0$.

constrained objective function, as summarized bellow:

---

**Algorithm 1    Nonnegative Sparse PCA (NSPCA)**

- *Start with an initial guess for $U$.*
- *Iterate over entries $(r, s)$ of $U$ until convergence:*
    - *Set the value of $u_{rs}$ to the global nonnegative maximizer of eqn. 4 by evaluating it over all nonnegative roots of eqn. 5 and zero.*

---

Caching some calculation results from the update of one element of $U$ to the other, each update is done in O($d$), and the entire matrix $U$ is updated in O($d^2 k$).

It is easy to see that the gradient at the convergence point of Alg. 1 is orthogonal to the constraints in eqn. 3, and therefore Alg. 1 converges to a local maximum of the problem. It is also worthwhile to compare this nonnegative coordinate-descent scheme with the nonnegative coordinate-descent scheme of Lee and Seung [10]. The update rule of [10] is multiplicative, which holds two inherent drawbacks. First, it cannot turn positive values into zero or vise versa, and therefore the solution will never be on the boundary itself, a drawback that does not exist in our scheme. Second, since it is multiplicative, the perseverance of nonnegativity is built upon the nonnegativity of the input, and therefore it cannot be applied to our problem while our scheme can be also applied to NMF. In other words, a practical aspect our the NSPCA algorithm is that it can handle general (not necessarily non-negative) input matrices — such as zero-mean covariance matrices.

## 5   Experiments

We start by demonstrating the role of the $\alpha$ and $\beta$ parameters in the task of extracting face parts. We use the MIT CBCL Face Dataset #1 of 2429 aligned face images, 19 by 19 pixels each, a dataset that was extensively used to demonstrate the ability of *Nonnegative Matrix Factorization (NMF)* [10] methods. We start with $\alpha = 2 \times 10^7$ and $\beta = 0$ to extract the 10 principal vectors in Fig. 2(a), and then increase $\alpha$ to $5 \times 10^8$ to get the principal vectors in Fig. 2(b). Note that as $\alpha$ increases the overlap among the principal vectors decreases and the holistic nature of some of the vectors in Fig. 2(a) vanishes. The vectors also become sparser, but this is only a byproduct of their non-overlapping nature. Fig. 3(a) shows the amount of overlap $\|I - U^T U\|$ as a function of $\alpha$, showing a consistence drop in the overlap as $\alpha$ increases. We now set $\alpha$ back to $2 \times 10^7$ as in Fig. 2(a), but set the value of $\beta$ to be $2 \times 10^6$ to get the factors in Fig. 2(d). The vectors become sparser as $\beta$ increases, but this time the sparseness emerges from a drop of less informative pixels within the original vectors of Fig. 2(a), rather than a replacement of the holistic principal vectors with ones that are part based in nature. The amount of non-zero elements in the principal vectors, $\|U\|_{L_0}$, is plotted as a function of $\beta$ in Fig. 3(b), showing the increment in sparseness as $\beta$ increases.

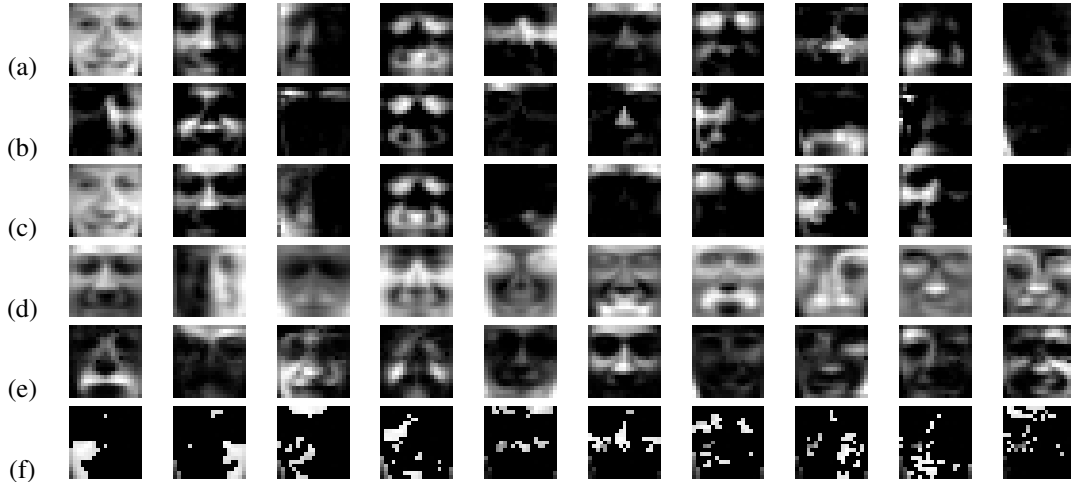

Figure 2: The role of $\alpha$ and $\beta$ is demonstrated in the task of extracting ten image features using the MIT-CBCL Face Dataset #1. At the top row (a), we use $\alpha = 2 \times 10^7$ and $\beta = 0$. In (b) we increase $\alpha$ to $5 \times 10^8$ while $\beta$ stays zero, to get more localized parts that has lower amount of overlap. In (c) we reset $\alpha$ to be $2 \times 10^7$ as in (a), but increase $\beta$ to be $2 \times 10^6$. While we increase $\beta$, pixels that explain less variance are dropped from the factors, but the overlapping nature of the factors remains. (See Fig. 3 for a detailed study.) In (d) we show the ten leading principal components of PCA, in (e) the ten factors of NMF, and in (f) the leading principal vectors of GSPCA when allowing 55 active pixels per principal vector.

Next we study how the different dimensional reduction methods aid the generalization ability of SVM in the task of face detection. To measure the generalization ability we use the *Receiver Operating Characteristics (ROC)* curve, a two dimensional graph measuring the classification ability of an algorithm over a dataset, showing the amount of true-positives as a function of the amount of false-positives. The wider the area under this curve is, the better the generalization is. Again, we use the MIT CBCL Face Dataset #1, where 1000 face images and 2000 non-face images were used as a training set, and the rest of the dataset used as a test set. The dimensional reduction was performed over the 1000 face images of the training set. We run linear SVM on the ten features extracted by NSPCA when using different values of $\alpha$ and $\beta$, showing in Fig. 4(a) that as the principal factors become less overlapping (higher $\alpha$) and sparser (higher $\beta$), the ROC curve is higher, meaning that SVM is able to generalize better. Next, we compare the ROC curve produced by linear SVM when using the NSPCA extracted features (with $\alpha = 5 \times 10^8$ and $\beta = 2 \times 10^6$) to the ones produced when using PCA and NMF (the principal vectors are displayed in Fig. 2(d) and Fig. 2(e), correspondingly). As a representative of the Sparse PCA methods we use the recent *Greedy Sparse PCA (GSPCA)* of [12] that shows comparable or better results to all other Sparse PCA methods (see the principal vectors in Fig. 2(f)). Fig. 4(b) shows that better generalization is achieved when using the NSPCA extracted features, and hence a more reliable face detection.

Since NSPCA is limited to nonnegative entries of the principal vectors, it can inherently explain less variance than Sparse PCA algorithms which are not constrained in that way, similarly to the fact that Sparse PCA algorithms can explain less variance than PCA. While this limitation holds, NSPCA still manages to explain a large amount of the variance. We demonstrate that in Fig. 5, where we compare the amount of cumulative explained variance and cumulative cardinality of different Sparse PCA algorithms over the Pit Props dataset, a classic dataset used throughout the Sparse PCA literature. In domains where nonnegativity is intrinsic to the problem, however, using NSPCA extracted features improves the generalization ability of learning algorithms, as we have demonstrated above for the face detection problem.

## 6  Summary

Our method differs substantially from previous approaches to sparse PCA — a difference that begins with the definition of the problem itself. Other sparse PCA methods try to limit the cardinality (number of non-zero elements) of each principal vector, and therefore accept as input a (soft) limitation on

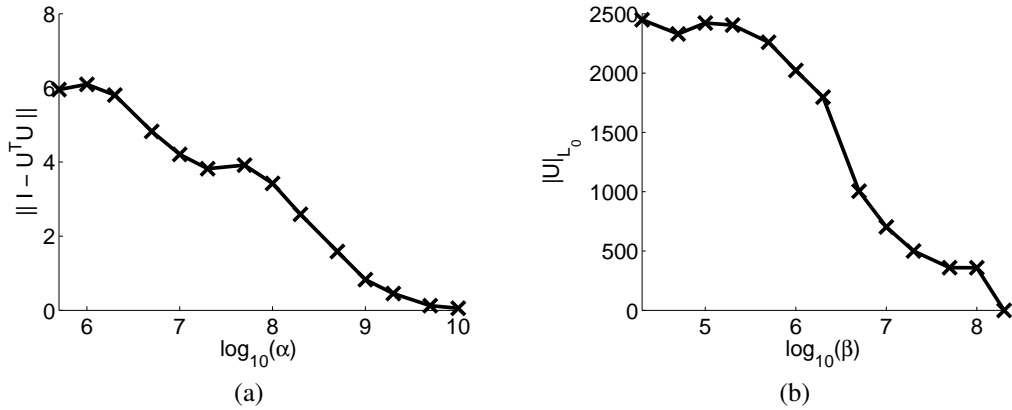

(a)                                                      (b)

Figure 3: (a) The amount of overlap and orthogonality as a function of $\alpha$, where higher values of $\alpha$ decrease the overlap and increase the orthogonality, and (b) the amount of non-zero elements as a function of $\beta$, where higher values of $\beta$ enforce sparseness.

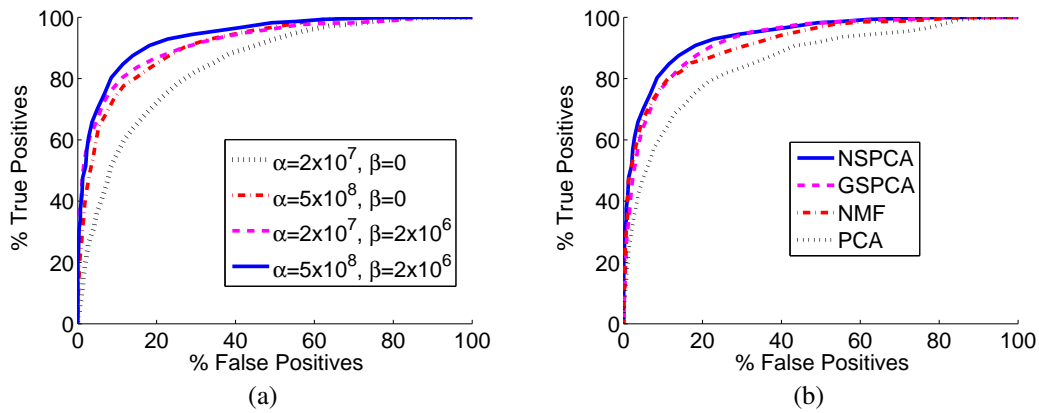

(a)                                                      (b)

Figure 4: The ROC curve of SVM in the task of face detection over the MIT CBCL Face Dataset #1 (a) when using different values of $\alpha$ and $\beta$, showing improved generalization when using principal vectors that has less overlap (higher $\alpha$) and that are sparser (higher $\beta$); and (b) when using NMF, PCA, GSPCA and NSPCA extracted features, showing better generalization when using NSPCA.

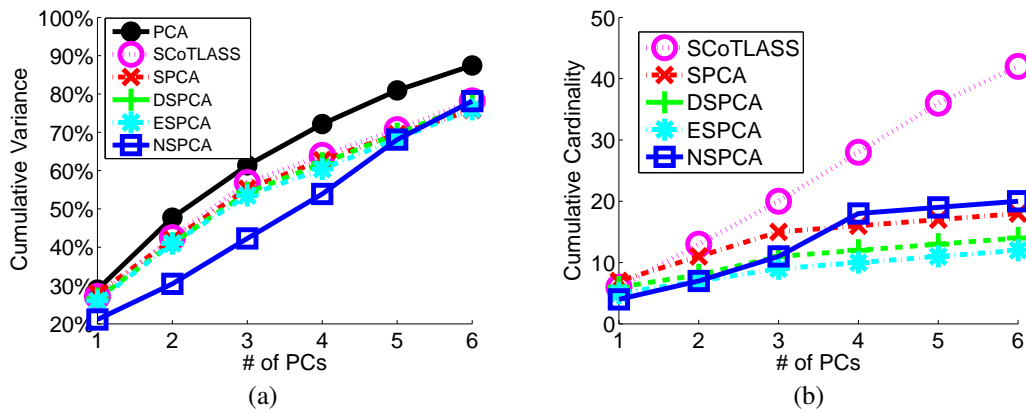

(a)                                                      (b)

Figure 5: (a) Cumulative explained variance and (b) cumulative cardinality as a function of the number of principal components on the Pit Props dataset, a classic dataset that is typically used to evaluate Sparse PCA algorithms. Although NSPCA is more constrained than other Sparse PCA algorithms, and therefore can explain less variance just like Sparse PCA algorithms can explain less variance than PCA, and although the dataset is not nonnegative in nature, NSPCA shows competitive results when the number of principal components increases.

that cardinality. In addition, most sparse PCA methods focus on the task of finding a single principal vector. Our method, on the other hand, splits the coordinates among the different principal vectors, and therefore its input is the number of principal vectors, or parts, rather than the size of each part. As a consequence, the natural way to use our algorithm is to search for all principal vectors together. In that sense, it bears resemblance to the Nonnegative Matrix Factorization problem, from which our method departs significantly in the sense that it focus on informative parts, as it maximizes the variance. Furthermore, the non-negativity of the output does not rely on having non-negative input matrices to the process thereby permitting zero-mean covariance matrices to be fed into the process just as being done with PCA.

## Footnotes

*School of Engineering and Computer Science, Hebrew University of Jerusalem, Jerusalem 91904, Israel.

## References

[1] Liviu Badea and Doina Tilivea. Sparse factorizations of gene expression guided by binding data. In *Pacific Symposium on Biocomputing*, 2005.

[2] Alexandre d'Aspremont, Laurent El Ghaoui, Michael I. Jordan, and Gert R. G. Lanckriet. A direct formulation for sparse PCA using semidefinite programming. In *Proceedings of the conference on Neural Information Processing Systems (NIPS)*, 2004.

[3] C. A. Floudas and V. Visweswaran. Quadratic optimization. In *Handbook of global optimization*, pages 217–269. Kluwer Acad. Publ., Dordrecht, 1995.

[4] Matthias Heiler and Christoph Schnörr. Learning non-negative sparse image codes by convex programming. In *Proc. of the 10th IEEE Intl. Conf. on Comp. Vision (ICCV)*, 2005.

[5] Patrik O. Hoyer. Non-negative sparse coding. In *Neural Networks for Signal Processing, 2002. Proceedings of the 2002 12th IEEE Workshop on*, pages 557–565, 2002.

[6] Patrik O. Hoyer. Non-negative matrix factorization with sparseness constraints. *Journal of Machine Learning Research*, 5:1457–1469, 2004.

[7] Ravi Jagannathan and Tongshu Ma. Risk reduction in large portfolios: Why imposing the wrong constraints helps. *Journal of Finance*, 58(4):1651–1684, 08 2003.

[8] Ian T. Jolliffe. Rotation of principal components: Choice of normalization constraints. *Journal of Applied Statistics*, 22(1):29–35, 1995.

[9] Ian T. Jolliffe, Nickolay T. Trendafilov, and Mudassir Uddin. A modified principal component technique based on the LASSO. *Journal of Computational and Graphical Statistics*, 12(3):531–547, September 2003.

[10] D. D. Lee and H. S. Seung. Learning the parts of objects by non-negative matrix factorization. *Nature*, 401(6755):788–791, October 1999.

[11] S. Li, X. Hou, H. Zhang, and Q. Cheng. Learning spatially localized, parts-based representation. In *Proceedings of the IEEE Conference on Computer Vision and Pattern Recognition*, 2001.

[12] Baback Moghaddam, Yair Weiss, and Shai Avidan. Spectral bounds for sparse pca: Exact and greedy algorithms. In *Proceedings of the conference on Neural Information Processing Systems (NIPS)*, 2005.

[13] Beresford N. Parlett. *The symmetric eigenvalue problem*. Prentice-Hall, Inc., Upper Saddle River, NJ, USA, 1980.

[14] H. Zou, T. Hastie, and R. Tibshirani. Sparse principal component analysis, 2004.
